# Latent Variable Models for Predicting File Dependencies in Large-Scale Software Development

**Diane J. Hu**[1], **Laurens van der Maaten**[1,2], **Youngmin Cho**[1], **Lawrence K. Saul**[1], **Sorin Lerner**[1]

[1]Dept. of Computer Science & Engineering, University of California, San Diego
[2]Pattern Recognition & Bioinformatics Lab, Delft University of Technology
{dhu,lvdmaaten,yoc002,saul,lerner}@cs.ucsd.edu

## Abstract

When software developers modify one or more files in a large code base, they must also identify and update other related files. Many file dependencies can be detected by mining the development history of the code base: in essence, groups of related files are revealed by the logs of previous workflows. From data of this form, we show how to detect dependent files by solving a problem in *binary* matrix completion. We explore different latent variable models (LVMs) for this problem, including Bernoulli mixture models, exponential family PCA, restricted Boltzmann machines, and fully Bayesian approaches. We evaluate these models on the development histories of three large, open-source software systems: *Mozilla Firefox*, *Eclipse Subversive*, and *Gimp*. In all of these applications, we find that LVMs improve the performance of related file prediction over current leading methods.

## 1 Introduction

As software systems grow in size and complexity, they become more difficult to develop and maintain. Nowadays, it is not uncommon for a code base to contain source files in multiple programming languages, text documents with meta information, XML documents for web interfaces, and even platform-dependent versions of the same application. This complexity creates many challenges because no single developer can be an expert in all things.

One such challenge arises whenever a developer wishes to update one or more files in the code base. Often, seemingly localized changes will require many parts of the code base to be updated. Unfortunately, these dependencies can be difficult to detect. Let $S$ denote a set of *starter* files that the developer wishes to modify, and let $R$ denote the set of *relevant* files that require updating after modifying $S$. In a large system, where the developer cannot possibly be familiar with the entire code base, automated tools that can recommend files in $R$ given starter files in $S$ are extremely useful.

A number of automated tools now make recommendations of this sort by mining the development history of the code base [1, 2]. Work in this area has been facilitated by code versioning systems, such as CVS or Subversion, which record the development histories of large software projects. In these histories, *transactions* denote sets of files that have been jointly modified—that is, whose changes have been submitted to the code base within a short time interval. Statistical analyses of past transactions can reveal which files depend on each other and need to be modified together.

In this paper, we explore the use of latent variable models (LVMs) for modeling the development history of large code bases. We consider a number of different models, including Bernoulli mixture models, exponential family PCA, restricted Boltzmann machines, and fully Bayesian approaches. In these models, the problem of recommending relevant files can be viewed as a problem in binary matrix completion. We present experimental results on the development histories of three large open-source systems: Mozilla Firefox, Eclipse Subversive, and Gimp. In all of these applications, we find that LVMs outperform the current leading method for mining development histories.

## 2 Related work

Two broad classes of methods are used for identifying file dependencies in large code bases; one analyzes the semantic content of the code base while the other analyzes its development history.

### 2.1 Impact analysis

The field of *impact analysis* [3] draws on tools from software engineering in order to identify the consequences of code modifications. Most approaches in this tradition attempt to identify program dependencies by inspecting and/or running the program itself. Such dependence-based techniques include transitive traversal of the call graph as well as static [4, 5, 6] and dynamic [7, 8] slicing techniques. These methods can identify many dependencies; however, they have trouble on certain difficult cases such as cross-language dependencies (e.g., between a data configuration file and the code that uses it) and cross-program dependencies (e.g., between the front and back ends of a compiler). These difficulties have led researchers to explore the methods we consider next.

### 2.2 Mining of development histories

Data-driven methods identify file dependencies in large software projects by analyzing their development histories. Two of the most widely recognized works in this area are by Ying et al. [1] and Zimmerman et al. [2]. Both groups use frequent itemset mining (FIM) [9], a general heuristic for identifying *frequent patterns* in large databases. The *patterns* extracted from development histories are just those sets of files that have been jointly modified at some point in the past; the *frequent patterns* are the patterns that have occurred at least $\tau$ times. The parameter $\tau$ is called the *minimum support threshold*. In practice, it is tuned to yield the best possible balance of precision and recall.

Given a database and a minimum support threshold, the resulting set of frequent patterns is uniquely specified. Much work has been devoted to making FIM as fast and efficient as possible. Ying et al. [1] uses a FIM algorithm called FP-growth, which extracts frequent patterns by using a tree-like data structure that is cleverly designed to prune the number of possible patterns to be searched. FP-growth is used to find all frequent patterns that contain the set of starter files; the joint sets of these frequent patterns are then returned as recommendations. As a baseline in our experiments we use a variant of FP-growth called FP-Max [10] which outputs only maximal sets for added efficiency.

Zimmerman et al. [2] uses the popular Apriori algorithm [11] (which uses FIM to solve a subtask) to form *association rules* from the development history. These rules are of the form $x_1 \rightarrow x_2$, where $x_1$ and $x_2$ are disjoint sets; they indicate that "if $x_1$ is observed, then based on experience, $x_2$ should also be observed." After identifying all rules in which starter files appear on the left hand side, their tool recommends all files that appear on the right hand side. They also work with content on a finer granularity, recommending not only relevant files, but also relevant code blocks within files.

Both Ying et al. [1] and Zimmerman et al. [2] evaluate the data-driven approach by its *f-measure*, as measured against "ground-truth" recommendations. For Ying et al. [1], these ground-truth recommendations are the files committed for a completed modification task, as recorded in that project's Bugzilla. For Zimmerman et al. [2], the ground-truth recommendations are the files checked-in together at some point in the past, as revealed by the development history.

Other researchers have also used the development history to detect file dependencies, but in markedly different ways. Shirabad et al. [12] formulate the problem as one of binary classification; they label pairs of source files as relevant or non-relevant based on their joint modification histories. Robillard [13] analyzes the topology of structural dependencies between files at the code-block level. Kagdi et al [14] improve on the accuracy of existing file recommendation methods by considering asymmetric file dependencies; this information is also used to return a partial ordering over recommended files. Finally, Sherriff et al. [15] identify clusters of dependent files by performing singular value decomposition on the development history.

## 3 Latent variable modeling of development histories

We examine four latent variable models of file dependence in software systems. All these models represent the development history as an $N \times D$ large binary matrix, where non-zero elements in

the same row indicate files that were checked-in together or jointly modified at some point in time. To detect dependent files, we infer the values of missing elements in this matrix from the values of known elements. The inferences are made from the probability distributions defined by each model. We use the following notation for all models:

1. The *file list* $\mathcal{F} = (f_1, \ldots, f_D)$ is an ordered collection of all files referenced in a static version of the development history.

2. A *transaction* is a set of files that were modified together, according to the development history. We represent each transaction by a $D$-dimensional binary vector $\mathbf{x} = (x_1, \ldots, x_D)$, where $x_i = 1$ if the $f_i$ is a member of the transaction, and $x_i = 0$ otherwise.

3. A *development history* $\mathcal{D}$ is a set of $N$ transaction vectors $\{\mathbf{x}_1, \mathbf{x}_2, \ldots, \mathbf{x}_N\}$. We assume them to be independently and identically sampled from some underlying joint distribution.

4. A *starter set* is a set of $s$ starter files $\mathcal{S} = (f_{i_1}, \ldots, f_{i_s})$ that the developer wishes to modify.

5. A *recommendation set* is a set of recommended files $\mathcal{R} = (f_{j_1}, \ldots, f_{j_r})$ that we label as relevant to the starter set $\mathcal{S}$.

## 3.1 Bernoulli mixture model

The simplest model that we explore is a Bernoulli mixture model (BMM). Figure 1(a) shows the BMM's graphical model in plate notation. In training, the observed variables are the $D$ binary elements $x_i \in \{0, 1\}$ of each transaction vector. The hidden variable is a multinomial label $z \in \{1, 2, \ldots, k\}$ that can be viewed as assigning each transaction vector to one of $k$ clusters. The joint distribution of the BMM is given by:

$$p(\mathbf{x}, z | \boldsymbol{\pi}, \boldsymbol{\mu}) \;=\; p(z|\boldsymbol{\pi}) \prod_{i=1}^{D} p(x_i | z, \boldsymbol{\mu}) \;=\; \pi_z \prod_{i=1}^{D} \mu_{iz}^{x_i} (1 - \mu_{iz})^{1-x_i}. \tag{1}$$

As implied by the graph in Fig. 1(a), we model the different elements of $\mathbf{x}$ as conditionally independent given the label $z$. Here, the parameter $\pi_z = p(z|\boldsymbol{\pi})$ denotes the prior probability of the latent variable $z$, while the parameter $\mu_{iz} = p(x_i = 1 | z, \boldsymbol{\mu})$ denotes the conditional mean of the observed variable $x_i$. We use the EM algorithm to estimate parameters that maximize the likelihood $p(\mathcal{D}|\boldsymbol{\pi}, \boldsymbol{\mu}) = \prod_n p(\mathbf{x}_n | \boldsymbol{\pi}, \boldsymbol{\mu})$ of the transactions in the development history.

When a software developer wishes to modify a set of starter files, she can query a trained BMM to identify a set of relevant files. Let $\mathbf{s} = \{x_{i_1}, \ldots, x_{i_s}\}$ denote the elements of the transaction vector indicating the files in the starter set $\mathcal{S}$. Let $\mathbf{r}$ denote the $D - s$ remaining elements of the transaction vector indicating files that may or may not be relevant. In BMMs, we infer which files are relevant by computing the posterior probability $p(\mathbf{r}|\mathbf{s} = \mathbf{1}, \boldsymbol{\pi}, \boldsymbol{\mu})$. Using Bayes rule and conditional independence, this posterior probability is given (up to a constant factor) by:

$$p(\mathbf{r}|\mathbf{s} = \mathbf{1}, \boldsymbol{\pi}, \boldsymbol{\mu}) \;\propto\; \sum_{z=1}^{k} p(\mathbf{r}|z, \boldsymbol{\mu}) \, p(\mathbf{s} = \mathbf{1} | z, \boldsymbol{\mu}) \, p(z|\boldsymbol{\pi}). \tag{2}$$

The most likely set of relevant files, according to the model, is given by the completed transaction $\mathbf{r}^*$ that maximizes the right hand side of eq. (2). Unfortunately, while we can efficiently compute the posterior probability $p(\mathbf{r}|\mathbf{s} = \mathbf{1})$ for a particular set of recommended files, it is not straightforward to maximize eq. (2) over all $2^{D-s}$ possible ways to complete the transaction. As an approximation, we sort the possibly relevant files by their individual posterior probabilities $p(x_i = 1 | \mathbf{s} = \mathbf{1})$ for $f_i \notin \mathcal{S}$. Then we recommend all files whose posterior probabilities $p(x_i = 1 | \mathbf{s} = \mathbf{1})$ exceed some threshold; we optimize the threshold on a held-out set of training examples.

## 3.2 Bayesian Bernoulli mixture model

We also explore a Bayesian treatment of the BMM. In a Bayesian Bernoulli mixture (BBM), instead of learning point estimates of the parameters $\{\boldsymbol{\pi}, \boldsymbol{\mu}\}$, we introduce a prior distribution $p(\boldsymbol{\pi}, \boldsymbol{\mu})$ and make predictions by averaging over the posterior distribution $p(\boldsymbol{\pi}, \boldsymbol{\mu}|\mathcal{D})$. The generative model for the BBM is shown graphically in Figure 1(b).

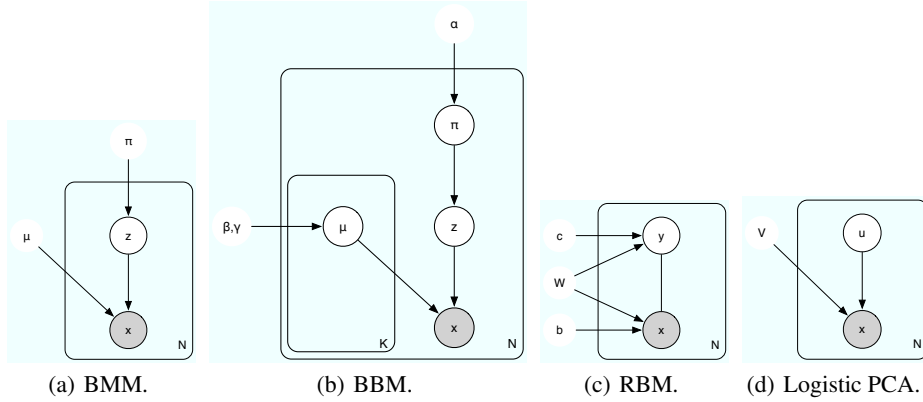

(a) BMM.    (b) BBM.    (c) RBM.    (d) Logistic PCA.

Figure 1: Graphical model of the Bernoulli mixture model (BMM), the Bayesian Bernoulli mixture (BBM), the restricted Boltzmann machine (RBM), and logistic PCA.

In our BBMs, the mixture weight parameters are drawn from a Dirichlet prior[1]:

$$p(\boldsymbol{\pi}|\alpha) = \text{Dirichlet}\left(\boldsymbol{\pi}\,|\alpha/k, \ldots, \alpha/k\right), \tag{3}$$

where $k$ indicates (as before) the number of mixture components and $\alpha$ is a hyperparameter of the Dirichlet prior, the so-called concentration parameter[2]. Likewise, the parameters of the $k$ Bernoulli distributions are drawn from Beta priors:

$$p(\boldsymbol{\mu}_j|\beta, \gamma) = \text{Beta}(\boldsymbol{\mu}_j|\beta, \gamma), \tag{4}$$

where $\boldsymbol{\mu}_j$ is a $D$-dimensional vector, and $\beta$ and $\gamma$ are hyperparameters of the Beta prior.

As exact inference in BBMs is intractable, we resort to collapsed Gibbs sampling and make predictions by averaging over samples from the posterior. In particular, we integrate out the Bernoulli parameters $\boldsymbol{\mu}$ and the cluster distribution parameters $\boldsymbol{\pi}$, and we sample the cluster assignment variables $\mathbf{z}$. For Gibbs sampling, we must compute the conditional probability $p(z_n = j|\mathbf{z}_{-n}, \mathcal{D})$ that the $n$th transaction is assigned to cluster $j$, given the training data $\mathcal{D}$ and all other cluster assignments $\mathbf{z}_{-n}$. This probability is given by:

$$p(z_n = j|\mathbf{z}_{-n}, \mathcal{D}) = \frac{N_{-nj} + \frac{\alpha}{k}}{N - 1 + \alpha} \prod_{i=1}^{D} \left[ \frac{(\beta + N_{-nij})^{x_{ni}}(\gamma + N_{-nj} - N_{-nij})^{(1-x_{ni})}}{\beta + \gamma + N_{-nj}} \right], \tag{5}$$

where $N_{-nj}$ counts the number of transactions assigned to cluster $j$ (excluding the $n$th transaction) and $N_{-nij}$ counts the number of times that the $i$th file belongs to one of these $N_{-nj}$ transactions.

After each full Gibbs sweep, we obtain a sample $\mathbf{z}^{(t)}$ (and corresponding counts $N_j^{(t)}$ of the number of points assigned to cluster $j$), which can be used to infer the Bernoulli parameters $\boldsymbol{\mu}_j^{(t)}$. We use $T$ of these samples to estimate the probability that a file $x_i$ needs to be changed given files in the starter set $\mathcal{S}$. In particular, averaging predictions over the $T$ Gibbs samples, we estimate:

$$p(x_i = 1|\mathbf{s} = \mathbf{1}) \approx \frac{1}{T} \sum_{t=1}^{T} \left[ \frac{1}{N} \sum_{j=1}^{k} N_j^{(t)} \frac{p\left(x_i = 1|\boldsymbol{\mu}_j^{(t)}\right)}{p\left(\mathbf{s} = \mathbf{1}|\boldsymbol{\mu}_j^{(t)}\right)} \right], \quad \text{with} \quad \boldsymbol{\mu}_j^{(t)} = \frac{1}{N_j^{(t)}} \sum_{n:z_n^{(t)}=j} \mathbf{x}_n. \tag{6}$$

### 3.3 Restricted Boltzmann Machines

A restricted Boltzmann machine (RBM) is a Markov random field (MRF) whose nodes are (typically) binary random variables [17]. The graphical model of an RBM is a fully connected bipartite

graph with $D$ observed variables $x_i$ in one layer and $k$ latent variables $y_j$ in the other; see Fig. 1(c). Due to the bipartite structure, the latent variables are conditionally independent given the observed variables (and vice versa). For the RBMs in this paper, we model the joint distribution as:

$$p(\mathbf{x}, \mathbf{y}) = \frac{1}{Z} \exp\left(-\mathbf{x}^\top \mathbf{W}\mathbf{y} - \mathbf{b}^\top \mathbf{x} - \mathbf{c}^\top \mathbf{y}\right), \tag{7}$$

where $\mathbf{W}$ stores the weight matrix between layers, $\mathbf{b}$ and $\mathbf{c}$ store (respectively) the biases on observed and hidden nodes, and $Z$ is a normalization factor that depends on the model's parameters. The product form of RBMs can model much sharper distributions over the observed variables than mixture models [17], making them an interesting alternative to consider for our application.

RBMs are trained by maximum likelihood estimation. Exact inference in RBMs is intractable due to the exponential sum in the normalization factor $Z$. However, the conditional distributions required for Gibbs sampling have a particularly simple form:

$$p(x_i = 1|\mathbf{y}) = \sigma\left(\sum_j W_{ij} y_j + \sum_j c_j\right), \tag{8}$$

$$p(y_j = 1|\mathbf{x}) = \sigma\left(\sum_i W_{ij} x_i + \sum_i b_i\right), \tag{9}$$

where $\sigma(z) = [1 + e^{-z}]^{-1}$ is the sigmoid function. The obtained Gibbs samples can be used to approximate the gradient of the likelihood function with respect to the model parameters; see [17, 18] for further discussion of sampling strategies[3].

To determine whether a file $f_i$ is relevant given starter files in $\mathcal{S}$, we can either (i) clamp the observed variables representing starter files and perform Gibbs sampling on the rest, or (ii) compute the posterior over the remaining files using a fast, factorized approximation [19]. In preliminary experiments, we found the latter to work best. Hence, we recommend files by computing

$$p(x_i = 1|\mathbf{s} = \mathbf{1}) \propto \exp(b_i) \prod_{\ell=1}^{k} \left(1 + \exp\left\{\sum_{j: f_j \in \mathcal{S}} x_j W_{j\ell} + W_{i\ell} + c_\ell\right\}\right), \tag{10}$$

then thresholding these probabilities on some value determined on held-out examples.

### 3.4 Logistic PCA

Logistic PCA is a method for dimensionality reduction of binary data; see Fig. 1(d) for its graphical model. Logistic PCA belongs to a family of algorithms known as exponential family PCA; these algorithms generalize PCA to data modeled by non-Gaussian distributions of the exponential family [20, 21, 22]. To use logistic PCA, we stack the $N$ transaction vectors $\mathbf{x}_n \in \{0, 1\}^D$ of the development history into a $N \times D$ binary matrix $\mathbf{X}$. Then, modeling each element of this matrix as a Bernoulli random variable, we attempt to find a low-rank factorization of the $N \times D$ *real-valued* matrix $\mathbf{\Theta}$ whose elements are the *log-odds* parameters of these random variables.

The low-rank factorization in logistic PCA is computed by maximizing the log-likelihood of the observed data $\mathbf{X}$. In terms of the log-odds matrix $\mathbf{\Theta}$, this log-likelihood is given by:

$$\mathcal{L}_{\mathbf{X}}(\mathbf{\Theta}) = \sum_{nd} \left[X_{nd} \log \sigma(\Theta_{nd}) + (1 - X_{nd}) \log \sigma(-\Theta_{nd})\right]. \tag{11}$$

We obtain a low dimensional representation of the data by factoring the log-odds matrix $\mathbf{\Theta} \in \Re^{N \times D}$ as the product of two smaller matrices $\mathbf{U} \in \Re^{N \times L}$ and $\mathbf{V} \in \Re^{L \times D}$. Specifically, we have:

$$\Theta_{nd} = \sum_\ell U_{n\ell} V_{\ell d}. \tag{12}$$

Note that the reduced rank $L \ll D$ plays a role analogous to the number of clusters $k$ in BMMs.

After obtaining a low-rank factorization of the log-odds matrix $\mathbf{\Theta} = \mathbf{UV}$, we can use it to recommend relevant files from starter files $\mathcal{S} = \{f_{i_1}, f_{i_2}, \ldots, f_{i_s}\}$. To recommend relevant files, we compute the vector $\mathbf{u}$ that optimizes the regularized log-loss:

$$\mathcal{L}_{\mathcal{S}}(\mathbf{u}) = \sum_{j=1}^{s} \log \sigma(\mathbf{u} \cdot \mathbf{v}_{i_j}) + \frac{\lambda}{2} \|\mathbf{u}\|^2, \tag{13}$$

| | Mozilla Firefox | | | Eclipse Subversive | | | Gimp | | |
|---|---|---|---|---|---|---|---|---|---|
| Time Period | March 2007 - Nov 2007 | | | Dec 2006 - May 2010 | | | Nov 2007 - May 2010 | | |
| Support | Train | Test | Files | Train | Test | Files | Train | Test | Files |
| 10 | 9,579 | 2,666 | 1,264 | 372 | 114 | 61 | 5,359 | 3,608 | 1,376 |
| 15 | 9,015 | 2,266 | 778 | 316 | 92 | 38 | 5,084 | 3,436 | 899 |
| 20 | 8,497 | 1,991 | 546 | 282 | 79 | 30 | 4,729 | 3,208 | 600 |
| 25 | 8,021 | 1,771 | 411 | 233 | 59 | 25 | 4,469 | 3,012 | 447 |

Table 1: Datasets statistics, showing the time period from which transactions were extracted, and the number of transactions and unique files in the training and test sets (for a single starter file).

where in the first term, $\mathbf{v}_\ell$ denotes the $\ell$th column of the matrix $\mathbf{V}$, and in the second term, $\lambda$ is a regularization parameter. The vector $\mathbf{u}$ obtained in this way is the low dimensional representation of the transaction with starter files in $\mathcal{S}$. To determine whether file $f_i$ is relevant, we compute the probability $p(x_i = 1|\mathbf{u}, \mathbf{V}) = \sigma(\mathbf{u} \cdot \mathbf{v}_i)$ and recommend the file if this probability exceeds some threshold. (We tune the threshold on held-out transactions from the development history).

## 4  Experiments

We evaluated our models on three datasets[4] constructed from check-in records of Mozilla Firefox, Eclipse Subversive, and Gimp. These open-source projects use software configuration management (SCM) tools which provide logs that allow us to extract binary vectors indicating which files were changed during a transaction. Our experimental setup and results are described below.

### 4.1  Experimental setup

We preprocess the raw data obtained from SCM's check-in records in two steps. First, following Ying et al [1], we eliminate all transactions consisting of more than 100 files (as these usually do not correspond to meaningful changes). Second, we simulate the *minimum support threshold* (see Section 2.2) by removing all files in the code base that occur very infrequently. This pruning allows us to make a fair comparison with latent variable models (LVMs).

After pre-processing, the dataset is chronologically ordered; the first two-thirds is used as training data, and the last one-third as testing data. For each transaction in the test set, we formed a "query" and "label" set by randomly picking a set of changed files as starter files. The remaining files that were changed in the transaction form the label set, which is the set of files our models must predict. Following [1], we only include transactions for which the label set is non-empty in the train data. Table 1 shows the number of transactions for training and test set, as well as the total number of unique files that appear in these transactions.

We trained the LVMs as follows. The Bernoulli mixture models (BMMs) were trained by 100 or fewer iterations of the EM algorithm. For the Bayesian mixtures (BBMs), we ran 30 separate Markov chains and made predictions after 30 full Gibbs sweeps[5]. The RBMs were trained for 300 iterations of contrastive divergence (CD), starting with CD-1 and gradually increasing the number of Gibbs sweeps to CD-9 [17]. The parameters $\mathbf{U}$ and $\mathbf{V}$ of logistic PCA were learned using an alternating least squares procedure [21] that converges to a local maximum of the log-likelihood. We initialized the matrices $\mathbf{U}$ and $\mathbf{V}$ from an SVD of the matrix $\mathbf{X}$.

The parameters of the LVMs (i.e., number of hidden components in the BMM and RBM, as well as the number of dimensions and the regularization parameter $\lambda$ in logistic PCA) were selected based on the performance on a small held-out validation set. The hyperparameters of the Bayesian Bernoulli mixtures were set based on prior knowledge from the domain: the Beta-prior parameters $\beta$ and $\gamma$ were set to $0.005$ and $0.95$, respectively, to reflect our prior knowledge that most files are not changed in a transaction. The concentration parameter $\alpha$ was set to $50$ to reflect our prior knowledge that file dependencies typically form a large number of small clusters.

| Model | Support | Mozilla Firefox | | | | Eclipse Subversive | | | | Gimp | | | |
|---|---|---|---|---|---|---|---|---|---|---|---|---|---|
| | | Start = 1 | | Start = 3 | | Start = 1 | | Start = 3 | | Start = 1 | | Start = 3 | |
| FIM | 10 | 0.106 | 0.136 | 0.112 | 0.195 | 0.133 | 0.382 | 0.234 | 0.516 | 0.020 | 0.116 | 0.016 | 0.176 |
| | 15 | 0.129 | 0.144 | 0.127 | 0.194 | 0.141 | 0.461 | 0.319 | 0.632 | 0.014 | 0.091 | 0.016 | 0.159 |
| | 20 | 0.115 | 0.137 | 0.106 | 0.186 | 0.177 | 0.550 | 0.364 | 0.672 | 0.007 | 0.066 | 0.013 | 0.129 |
| | 25 | 0.124 | 0.135 | 0.110 | 0.195 | 0.227 | 0.616 | 0.360 | 0.637 | 0.006 | 0.057 | 0.010 | 0.095 |
| BMM | 10 | 0.160 | 0.189 | 0.106 | 0.158 | 0.222 | 0.433 | 0.206 | 0.479 | 0.129 | 0.177 | 0.084 | 0.152 |
| | 15 | 0.160 | 0.202 | 0.110 | 0.141 | 0.181 | 0.486 | 0.350 | 0.489 | **0.134** | 0.205 | 0.085 | 0.143 |
| | 20 | 0.172 | 0.204 | 0.120 | 0.147 | 0.196 | 0.530 | 0.403 | 0.514 | 0.127 | 0.207 | 0.085 | 0.154 |
| | 25 | 0.177 | 0.218 | 0.130 | 0.160 | 0.251 | 0.566 | 0.382 | 0.482 | 0.117 | 0.212 | 0.010 | 0.131 |
| BBM | 10 | 0.196 | 0.325 | 0.180 | 0.376 | 0.257 | 0.547 | 0.278 | 0.700 | 0.114 | 0.174 | 0.104 | 0.177 |
| | 15 | 0.192 | 0.340 | 0.180 | 0.376 | 0.202 | 0.607 | 0.374 | 0.769 | 0.114 | 0.200 | 0.107 | 0.183 |
| | 20 | **0.206** | 0.355 | **0.191** | **0.417** | 0.223 | 0.655 | 0.413 | **0.791** | 0.114 | 0.205 | 0.108 | 0.187 |
| | 25 | 0.197 | **0.360** | 0.175 | 0.391 | **0.262** | **0.694** | **0.418** | 0.756 | 0.110 | 0.206 | 0.103 | 0.179 |
| RBM | 10 | 0.157 | 0.230 | 0.069 | 0.307 | 0.170 | 0.233 | 0.090 | 0.405 | 0.074 | 0.137 | 0.028 | 0.194 |
| | 15 | 0.156 | 0.246 | 0.063 | 0.310 | 0.157 | 0.238 | 0.138 | 0.423 | 0.080 | 0.148 | 0.024 | 0.205 |
| | 20 | 0.169 | 0.260 | 0.058 | 0.324 | 0.174 | 0.307 | 0.178 | 0.531 | 0.074 | 0.156 | 0.027 | 0.242 |
| | 25 | 0.172 | 0.269 | 0.088 | 0.340 | 0.200 | 0.426 | 0.259 | 0.524 | 0.062 | 0.143 | 0.025 | 0.230 |
| LPCA | 10 | 0.200 | 0.249 | 0.169 | 0.300 | 0.124 | 0.415 | 0.230 | 0.609 | 0.123 | 0.187 | **0.148** | 0.263 |
| | 15 | 0.182 | 0.254 | 0.157 | 0.295 | 0.138 | 0.452 | 0.281 | 0.615 | 0.124 | 0.200 | 0.145 | **0.288** |
| | 20 | 0.182 | 0.265 | 0.156 | 0.308 | 0.212 | 0.517 | 0.325 | 0.667 | 0.115 | **0.222** | 0.135 | 0.300 |
| | 25 | 0.174 | 0.277 | 0.162 | 0.325 | 0.247 | 0.605 | 0.344 | 0.625 | 0.100 | 0.205 | 0.131 | 0.230 |

Table 2: Performance of FIM and LVMs on three datasets for queries with 1 or 3 starter files. Each shaded column presents the $f$-measure, and each white column presents the correct prediction ratio.

## 4.2 Results

Our experiments evaluated the performance of each LVM, as well as a highly efficient implementation of FIM called FP-Max [10]. Several experiments were run on different values of starter files (abbreviated "Start") and minimum support thresholds (abbreviated "Support"). Table 2 shows the comparison of each model in terms of the $f$-measure (the harmonic mean of the precision and recall) and the "correct prediction ratio," or CPR (the fraction of files we predict correctly, assuming that the number of files to be predicted is given). The latter measure reflects how well our models identify relevant files for a particular starter file, without the added complication of thresholding. Experiments that achieve the highest result for each of the two measures are boldfaced.

From our results, we see that most LVMs outperform the popular FIM approach. In particular, the BBMs outperform all other approaches on two of the three datasets, with a high of CPR = 79% in Eclipse Subversive. This means that an average of 79% of all dependent files are detected as relevant by the BBM. We also observe that $f$-measure generally decreases with the addition of starter files – since the average size of transactions is relatively small (around four files for Firefox), adding starter files must make predictions less obvious in the case that the total number of relevant files is not given to us. Increasing support, on the other hand, seems to effectively remove noise caused by infrequent files. Finally, we see that recommendations are most accurate on Eclipse Subversive, the smallest dataset. We believe this is because a smaller test set does not require a model to predict as far into the future as a larger one. Thus, our results suggest that an online learning algorithm may further increase accuracy.

## 5 Discussion

The use of LVMs has significant advantages over traditional approaches to impact analysis (see Section 2), namely its ability to find dependent files written in different languages. To show this, we present the three clusters with the highest weights, as discovered by a BMM in the Firefox data, in Table 3. The table reveals that the clusters correspond to interpretable structure in the code that span multiple data formats and languages. The first cluster deals with the JIT compiler for JavaScript, while the second and third deal with the CSS style sheet manager and web browser properties. The dependencies in the last two clusters would have been missed by conventional impact analysis.

| Cluster 1 | Cluster 2 | Cluster 3 |
|---|---|---|
| js/src/jscntxt.h | view/src/nsViewManager.cpp | browser/base/content/browser-context.inc |
| js/src/jstracer.cpp | layout/generic/nsHTMLReflowState.cpp | browser/base/content/browser.js |
| js/src/nanojit/Assembler.cpp | layout/reftests/bugs/reftest.list | browser/base/content/pageinfo/pageInfo.xul |
| js/src/jsregexp.cpp | layout/style/nsCSSRuleProcessor.cpp | browser/locales/en-US/chrome/browser/browser.dtd |
| js/src/jsapi.cpp | layout/style/nsCSSStyleSheet.cpp | toolkit/mozapps/update/src/nsUpdateService.js.in |
| js/src/jsarray.cpp | layout/style/nsCSSParser.cpp | toolkit/mozapps/update/src/updater/updater.cpp |
| js/src/jsfun.cpp | layout/base/crashtests/crashtests.list | modules/plugin/base/src/nsNPAPIPluginInstance.h |
| js/src/jsinterp.cpp | layout/base/nsBidiPresUtils.cpp | modules/plugin/base/src/nsPluginHost.cpp |
| js/src/jsnum.cpp | layout/base/nsPresShell.cpp | browser/locales/en-US/chrome/browser/browser.properties |
| js/src/jsobj.cpp | content/xbl/src/nsBindingManager.cpp | view/src/nsViewManager.cpp |

Table 3: Three of the clusters from Firefox, identified by the BMM. We show the clusters with the largest mixing proportion. Within each cluster, the 10 files with highest membership probabilities are shown; note how these files span multiple data formats and program languages, revealing dependencies that would escape the notice of traditional methods.

LVMs also have important advantages over FIM. Given a set $\mathcal{S}$ of starter files, FIM simply looks at co-occurrence data; it recommends a set of files $\mathcal{R}$ for which the number of transactions that contain both $\mathcal{R}$ and $\mathcal{S}$ is *frequent*. By contrast, LVMs can exploit higher-order information by discovering the underlying structure of the data. Our results suggest that the ability to leverage such structure leads to better predictions. Admittedly, in terms of computation, LVMs have a larger one-time training cost than the FIM, as we must first train the model or generate and store the Gibbs samples. However, for a single query, the time required to compute recommendations is comparable to that of the FP-Max algorithm we used for FIM.

The results from the previous section also revealed significant differences between the LVMs we considered. In the majority of our experiments, mixture models (with many mixture components) appear to outperform RBMs and logistic PCA. This result suggests that our dataset consists of a large number of transactions with a number of small, highly interrelated files. Modeling such data with a product of experts such as an RBM is difficult as each individual expert has the ability to "veto" a prediction. We tried to resolve this problem by using a sparsity prior on the states of the hidden units **y** to make the RBMs behave more like a mixture model [23], but in preliminary experiments, we did not find this to improve the performance. Another interesting observation is that the Bayesian treatment of the Bernoulli mixture model generally leads to better predictions than a maximum likelihood approach, as it is less susceptible to overfitting. This advantage is particularly useful in file dependency prediction which requires models with a large number of mixture components to appropriately model data that consists of many small, distinct clusters while having few training instances (i.e., transactions).

# 6 Conclusion

In this paper, we have described a new application of binary matrix completion for predicting file dependencies in software projects. For this application, we investigated the performance of four different LVMs and compared our results to that of the widely used of FIM. Our results indicate that LVMs can significantly outperform FIM by exploiting latent, higher-order structure in the data.

Admittedly, our present study is still limited in scope, and it is very likely that our results can be further improved. For instance, results from the Netflix competition have shown that blending the predictions from various models often leads to better performance [24]. The raw transactions also contain additional information that could be harvested to make more accurate predictions. Such information includes the identity of users who committed transactions to the code base, as well as the text of actual changes to the source code. It remains a grand challenge to incorporate all the available information from development histories into a probabilistic model for predicting which files need to be modified. In future work, we aim to explore discriminative methods for parameter estimation, as well as online algorithms for tracking non-stationary trends in the code base.

**Acknowledgments**

LvdM acknowledges support by the Netherlands Organisation for Scientific Research (grant no. 680.50.0908) and by EU-FP7 NoE on Social Signal Processing (SSPNet).

## Footnotes

[1]In preliminary experiments, we also investigated an infinite mixture of Bernoulli distributions that replaces the Dirichlet prior by a Dirichlet process [16]. However, we did not find the infinite mixture model to outperform its finite counterpart, so we do not discuss it further.

[2]For simplicity, we assume a symmetric Dirichlet prior, i.e. we assume $\forall j : \alpha_j = \alpha/k$.

[3]We use the approach in [17] known as contrastive divergence with $m$ Gibbs sweeps (CD-$m$).

[4]These binary datasets publicly available at `http://cseweb.ucsd.edu/~dhu/research/msr`

[5]In preliminary experiments, we found 30 Gibbs sweeps to be sufficient for the Markov chain to mix.

# References

[1] A.T.T. Ying, G.C. Murphy, R. Ng, and M.C. Chu-Carroll. Predicting source code changes by mining change history. *IEEE Transactions on Software Engineering*, 30(9):574–586, 2004.

[2] T. Zimmerman, P. Weibgerber, S. Diehl, and A. Zeller. Mining version histories to guide software changes. *Proceedings of the $26^{th}$ International Conference on Software Engineering*, pages 563–572, 2004.

[3] R. Arnold and S. Bohner. *Software Change Impact Analysis*. IEEE Computer Society, 1996.

[4] M. Weiser. Program slicing. In *Proceedings of the $5^{th}$ International Conference on Software Engineering*, pages 439–449, 1981.

[5] S. Horwitz, T. Reps, and D. Binkley. Interprocedural slicing using dependence graphs. *ACM Transactions on Programming Languages and Systems*, 12(1):26–60, 1990.

[6] F. Tip. A survey of program slicing techniques. *Journal of Programming Languages*, 3:121–189, 1995.

[7] B. Korel and J. Laski. Dynamic program slicing. *Information Processing Letters*, 29(3):155–163, 1988.

[8] X. Zhang, R. Gupta, and Y. Zhang. Precise dynamic slicing algorithms. In *Proceedings of the $25^{th}$ International Conference on Software Engineering*, pages 319–329, 2003.

[9] B. Goethals. Frequent set mining. In *The Data Mining and Knowledge Discovery Handbook*, pages 377–397, 2005.

[10] G. Grahne and J. Zhu. Efficiently using prefix-trees in mining frequent itemsets. *Proceedings of the $1^{st}$ ICDM Workshop on Frequent Itemset Mining Implementations*, 2003.

[11] M.J. Zaki, S. Parthasarathy, M. Ogihara, and W. Li. *New algorithms for fast discovery of association rules*. 1997.

[12] J. S. Shirabad, T. C. Lethbridge, and S. Matwin. Mining the maintenance history of a legacy software system. *Proceedings of the $19^{th}$ International Conference on Software Maintenance*, pages 95–104, 2003.

[13] M. Robillard. Automatic generation of suggestions for program investigation. *ACM SIGSOFT International Symposium on Foundations of Software Engineering*, 30:11–20, 2005.

[14] H. Kagdi, S. Yusaf, and J.I. Maletic. Mining sequences of changed-files from version histories. *Proc. of Int. Workshop on Mining Software Repositories*, pages 47–53, 2006.

[15] M. Sherriff, J.M. Lake, and L. Williams. Empirical software change impact analysis using singular value decomposition. *International Conference on Software Testing, Verification, and Validation*, 2008.

[16] R.M. Neal. Markov chain sampling methods for Dirichlet process mixture models. *Journal of Computational and Graphical Statistics*, 9:249–265, 2000.

[17] G.E. Hinton. Training products of experts by minimizing contrastive divergence. *Neural Computation*, 14(8):1771–1800, 2002.

[18] T. Tieleman. Training Restricted Boltzmann Machines using approximations to the likelihood gradient. In *Proceedings of the International Conference on Machine Learning*, volume 25, pages 1064–1071, 2008.

[19] R.R. Salakhutdinov, A. Mnih, and G.E. Hinton. Restricted Boltzmann Machines for collaborative filtering. In *Proceedings of the $24^{th}$ International Conference on Machine Learning*, pages 791–798, 2007.

[20] M. Collins, S. Dasgupta, and R.E. Schapire. A generalization of principal components analysis to the exponential family. In T. G. Dietterich, S. Becker, and Z. Ghahramani, editors, *Advances in Neural Information Processing Systems 14*, Cambridge, MA, 2002. MIT Press.

[21] A.I. Schein, L.K. Saul, and L.H. Ungar. A generalized linear model for principal component analysis of binary data. In *Proceedings of the $9^{th}$ International Workshop on Artificial Intelligence and Statistics*, 2003.

[22] I. Rish, G. Grabarnik, G. Cecchi, F. Pereira, and G.J. Gordon. Closed-form supervised dimensionality reduction with generalized linear models. In *Proceedings of the $25^{th}$ International Conference on Machine learning*, pages 832–839, 2008.

[23] M.A. Ranzato, Y.L. Boureau, and Y. LeCun. Sparse feature learning for deep belief networks. In *Advances in Neural Information Processing Systems*, pages 1185–1192, 2008.

[24] R.M. Bell and Y. Koren. Lessons from the Netflix prize challenge. *ACM SIGKDD Explorations Newsletter*, 9(2):75–79, 2007.

